# Cluster Stability for Finite Samples

**Ohad Shamir**[†] **and Naftali Tishby**[†‡]
† School of Computer Science and Engineering
‡ Interdisciplinary Center for Neural Computation
The Hebrew University
Jerusalem 91904, Israel
{ohadsh,tishby}@cs.huji.ac.il

## Abstract

Over the past few years, the notion of stability in data clustering has received growing attention as a cluster validation criterion in a sample-based framework. However, recent work has shown that as the sample size increases, any clustering model will usually become asymptotically stable. This led to the conclusion that stability is lacking as a theoretical and practical tool. The discrepancy between this conclusion and the success of stability in practice has remained an open question, which we attempt to address. Our theoretical approach is that stability, as used by cluster validation algorithms, is similar in certain respects to measures of generalization in a model-selection framework. In such cases, the model chosen governs the convergence rate of generalization bounds. By arguing that these rates are more important than the sample size, we are led to the prediction that stability-based cluster validation algorithms should not degrade with increasing sample size, despite the asymptotic universal stability. This prediction is substantiated by a theoretical analysis as well as some empirical results. We conclude that stability remains a meaningful cluster validation criterion over finite samples.

## 1 Introduction

Clustering is one of the most common tools of unsupervised data analysis. Despite its widespread use and an immense amount of literature, distressingly little is known about its theoretical foundations [14]. In this paper, we focus on sample based clustering, where it is assumed that the data to be clustered are actually a sample from some underlying distribution.

A major problem in such a setting is assessing cluster validity. In other words, we might wish to know whether the clustering we have found actually corresponds to a meaningful clustering of the underlying distribution, and is not just an artifact of the sampling process. This problem relates to the issue of model selection, such as determining the number of clusters in the data or tuning parameters of the clustering algorithm. In the past few years, cluster stability has received growing attention as a criterion for addressing this problem. Informally, this criterion states that if the clustering algorithm is repeatedly applied over independent samples, resulting in 'similar' clusterings, then these clusterings are statistically significant. Based on this idea, several cluster validity methods have been proposed (see [9] and references therein), and were shown to be relatively successful for various data sets in practice.

However, in recent work, it was proven that under mild conditions, stability is asymptotically fully determined by the behavior of the objective function which the clustering algorithm attempts to optimize. In particular, the existence of a unique optimal solution for some model choice implies stability as sample size increase to infinity. This will happen regardless of the model fit to the data. From this, it was concluded that stability is not a well-suited tool for model selection in clustering. This left open, however, the question of why stability is observed to be useful in practice.

In this paper, we attempt to explain why stability measures should have much wider relevance than what might be concluded from these results. Our underlying approach is to view stability as a measure of generalization, in a learning-theoretic sense. When we have a 'good' model, which is stable over independent samples, then inferring its fit to the underlying distribution should be easy. In other words, stability should 'work' because stable models generalize better, and models which generalize better should fit the underlying distribution better. We emphasize that this idea in itself is not novel, appearing explicitly and under various guises in many aspects of machine learning. The novelty in this paper lies mainly in the predictions that are drawn from it for clustering stability.

The viewpoint above places emphasis on the nature of stability for *finite* samples. Since generalization is meaningless when the sample is infinite, it should come as no surprise that stability displays similar behavior. On finite samples, the generalization uncertainty is virtually always strictly positive, with different model choices leading to different convergence rates towards zero for increasing sample size. Based on the link between stability and generalization, we predict that on realistic data, all risk-minimizing models asymptotically become stable, but the *rates of convergence* to this ultimate stability differ. In other words, an appropriate scaling of the stability measures will make them independent of the actual sample size used. Using this intuition, we characterize and prove a mild set of conditions, applicable in principle to a wide class of clustering settings, which ensure the relevance of cluster stability for arbitrarily large sample sizes. We then prove that the stability measure used in previous work to show negative asymptotic results on stability, actually allows us to discern the 'correct' model, regardless of how large is the sample, for a certain simple setting. Our results are further validated by some experiments on synthetic and real world data.

## 2 Definitions and notation

We assume that the data sample to be clustered, $S = \{\mathbf{x}_1, .., \mathbf{x}_m\}$, is produced by sampling instances i.i.d from an underlying distribution $\mathcal{D}$, supported on a subset $\mathcal{X}$ of $\mathbb{R}^n$. A *clustering* $C_D$ for some $D \subseteq \mathcal{X}$ is a function from $D \times D$ to $\{0, 1\}$, defining an equivalence relation on $D$ with a finite number of equivalence classes (namely, $C_D(\mathbf{x}_i, \mathbf{x}_j) = 1$ if $\mathbf{x}_i$ and $\mathbf{x}_j$ belong to the same cluster, and 0 otherwise). For a clustering $C_{\mathcal{X}}$ of the instance space, and a finite sample $S$, let $C_{\mathcal{X}}|_S$ denote the functional restriction of $C_{\mathcal{X}}$ on $S \times S$.

A *clustering algorithm* $A$ is a function from any finite sample $S \subseteq \mathcal{X}$, to some clustering $C_{\mathcal{X}}$ of the instance space[1]. We assume the algorithm is driven by optimizing an objective function, and has some user-defined parameters $\Theta$. In particular, $A_k$ denotes the algorithm $A$ with the number of clusters chosen to be $k$.

Following [2], we define the *stability* of a clustering algorithm $A$ on finite samples of size $m$ as:

$$stab(A, \mathcal{D}, m) \;=\; \mathbb{E}_{S_1, S_2} d_{\mathcal{D}}(A(S_1), A(S_2)), \tag{1}$$

where $S_1$ and $S_2$ are samples of size $m$, drawn i.i.d from $\mathcal{D}$, and $d_{\mathcal{D}}$ is some 'dissimilarity' function between clusterings of $\mathcal{X}$, to be specified later.

Let $\ell$ denote a *loss function* from any clustering $C_S$ of a finite set $S \subseteq \mathcal{X}$ to $[0, 1]$. $\ell$ may or may not correspond to the objective function the clustering algorithm attempts to optimize, and may involve a global quality measure rather than some average over individual instances. For a fixed sample size, we say that $\ell$ obeys the *bounded differences property* (see [11]), if for any clustering $C_S$ it holds that $|\ell(C_S) - \ell(C_{S'})| \leq a$, where $a$ is a constant, and $C_{S'}$ is obtained from $C_S$ by replacing at most one instance of $S$ by any other instance from $\mathcal{X}$, and clustering it arbitrarily.

A *hypothesis class* $H$ is defined as some set of clusterings of $\mathcal{X}$. The *empirical risk* of a clustering $C_{\mathcal{X}} \in H$ on a sample $S$ of size $m$ is $\ell(C_{\mathcal{X}}|_S)$. The *expected risk* of $C_{\mathcal{X}}$, with respect to samples $S$ of size $m$, will be defined as $\mathbb{E}_S \ell(C_{\mathcal{X}}|_S)$. The problem of generalization is how to estimate the expected risk, based on the empirical data.

# 3  A Bayesian framework for relating stability and generalization

The relationship between generalization and various notions of stability is long known, but has been dealt with mostly in a supervised learning setting (see [3][5] [8] and references therein). In the context of unsupervised data clustering, several papers have explored the relevance of statistical stability and generalization, separately and together (such as [1][4][14][12]). However, there are not many theoretical results quantitatively characterizing the relationship between the two in this setting. The aim of this section is to informally motivate our approach, of viewing stability and generalization in clustering as closely related.

Relating the two is very natural in a Bayesian setting, where clustering stability implies an 'unsurprising' posterior given a prior, which is based on clustering another sample. Under this paradigm, we might consider 'soft clustering' algorithms which return a distribution over a measurable hypothesis class $H$, rather than a specific clustering. This distribution typically reflects the likelihood of a clustering hypothesis, given the data and prior assumptions. Extending our notation, we have that for any sample $S$, $A(S)$ is now a distribution over $H$. The empirical risk of such a distribution, with respect to sample $S'$, is defined as $\ell(A(S)|_{S'}) = \mathbb{E}_{C_{\mathcal{X}} \sim A(S)} \ell(C_{\mathcal{X}}|_{S'})$.

In this setting, consider for example the following simple procedure to derive a clustering hypothesis distribution, as well as a generalization bound: Given a sample of size $2m$ drawn i.i.d from $\mathcal{D}$, we randomly split it into two samples $S_1, S_2$ each of size $m$, and use $A$ to cluster each of them separately. Then we have the following:

**Theorem 1.** *For the procedure defined above, assume $\ell$ obeys the bounded differences property with parameter $1/m$. Define the clustering distance $d_{\mathcal{D}}(\mathcal{P}, \mathcal{Q})$ in Eq. (1), between two distributions $\mathcal{P}, \mathcal{Q}$ over the hypothesis class $H$, as the Kullback-Leibler divergence $D_{KL}[\mathcal{Q}||\mathcal{P}]^2$. Then for a fixed confidence parameter $\delta \in (0,1)$, it holds with probability at least $1 - \delta$ over the draw of samples $S_1$ and $S_2$ of size $m$, that*

$$\mathbb{E}_S \ell(A(S_2)|_S) - \ell(A(S_2)|_{S_2}) \leq \sqrt{\frac{d_{\mathcal{D}}(A(S_1), A(S_2)) + \ln(m/\delta) + 2}{2m - 1}}.$$

The theorem is a straightforward variant of the PAC-Bayesian theorem [10]. Since the loss function is not necessarily an empirical average, we need to utilize McDiarmid's bound for random variables with bounded differences, instead of Hoeffding's bound. Other than that, the proof is identical, and is therefore ommited.

This theorem implies that the more stable is the Bayesian algorithm, the tighter the expected generalization bounds we can achieve. In fact, the 'expected' magnitude of the high-probability bound we will get (over drawing $S_1$ and $S_2$ and performing the procedure described above) is:

$$\mathbb{E}_{S_1, S_2} \sqrt{\frac{d_{\mathcal{D}}(A(S_1), A(S_2)) + ln(m/\delta) + 2}{2m - 1}} \quad \leq \quad \sqrt{\frac{\mathbb{E}_{S_1, S_2} d_{\mathcal{D}}(A(S_1), A(S_2)) + \ln(m/\delta) + 2}{2m - 1}}$$

$$= \sqrt{\frac{stab(A, \mathcal{D}, m) + \ln(m/\delta) + 2}{2m - 1}}.$$

Note that the only model-dependent quantity in the expression above is $stab(A, \mathcal{D}, m)$. Therefore, carrying out model selection by attempting to minimize these types of generalization bounds is closely related to minimizing $stab(A, \mathcal{D}, m)$. In general, the generalization bound might converge to 0 as $m \to \infty$, but this is immaterial for the purpose of model selection. The important factor is the relative values of the measure, over different choices of the algorithm parameters $\Theta$. In other words, the important quantity is the relative convergence rates of this bound for different choices of $\Theta$, governed by $stab(A, \mathcal{D}, m)$.

This informal discussion only exemplifies the relationship between generalization and stability, since the setting and the definition of $d_{\mathcal{D}}$ here differs from the one we will focus on later in the paper. Although these ideas can be generalized, they go beyond the scope of this paper, and we leave it for future work.

## 4 Effective model selection for arbitrarily large sample sizes

From now on, following [2], we will define the clustering distance function $d_{\mathcal{D}}$ of Eq. (1) as:

$$d_{\mathcal{D}}(A(S_1), A(S_2)) = \Pr_{\mathbf{x}_1, \mathbf{x}_2 \sim \mathcal{D}} (A(S_1)(\mathbf{x}_1, \mathbf{x}_2) \neq A(S_2)(\mathbf{x}_1, \mathbf{x}_2)). \tag{2}$$

In other words, the clustering distance is the probability that two independently drawn instances from $\mathcal{D}$ will be in the same cluster under one clustering, and in different clusters under another clustering.

In [2], it is essentially proven that if there exists a unique optimizer to the clustering algorithm's objective function, to which the algorithm converges for asymptotically large samples, then $stab(A, \mathcal{D}, m)$ converges to 0 as $m \to \infty$, regardless of the parameters of $A$. From this, it was concluded that using stability as a tool for cluster validity is problematic, since for large enough samples it would always be approximately zero, for any algorithm parameters chosen.

However, using the intuition gleaned from the results of the previous section, the different *convergence rates* of the stability measure (for different algorithm parameters) should be more important than their absolute values or the sample size. The key technical result needed to substantiate this intuition is the following theorem:

**Theorem 2.** *Let $X, Y$ be two random variables bounded in $[0, 1]$, and with strictly positive expected values. Assume $\mathbb{E}[X]/\mathbb{E}[Y] \geq 1 + c$ for some positive constant $c$. Letting $X_1, \ldots, X_m$ and $Y_1, \ldots, Y_m$ be $m$ identical independent copies of $X$ and $Y$ respectively, define $\hat{X} = \frac{1}{m}\sum_{i=1}^{m} X_i$ and $\hat{Y} = \frac{1}{m}\sum_{i=1}^{m} Y_i$. Then it holds that:*

$$\Pr(\hat{X} \leq \hat{Y}) \leq \exp\left(-\frac{1}{8} m \mathbb{E}[X]\left(\frac{c}{1+c}\right)^4\right) + \exp\left(-\frac{1}{4} m \mathbb{E}[X]\left(\frac{c}{1+c}\right)^2\right).$$

The importance of this theorem becomes apparent when $\hat{X}, \hat{Y}$ are taken to be empirical estimators of $stab(A, \mathcal{D}, m)$ for two different algorithm parameter sets $\Theta, \Theta'$. For example, suppose that according to our stability measure (see Eq. (1)), a cluster model with $k$ clusters is more stable than a model with $k'$ clusters, where $k \neq k'$, for sample size $m$ (e.g., $stab(A_k, \mathcal{D}, m) < stab(A_{k'}, \mathcal{D}, m)$). These stability measures might be arbitrarily close to zero. Assume that with high probability over the choice of samples $S_1$ and $S_2$ of size $m$, we can show that $d_{\mathcal{D}}(A_k(S_1), A_k(S_2)) \leq 1/\sqrt{m}$, while $d_{\mathcal{D}}(A_{k'}(S_1), A_{k'}(S_2)) \geq 1.01/\sqrt{m}$. We cannot compute these exactly, since the definition of $d_{\mathcal{D}}$ involves an expectation over the unknown distribution $\mathcal{D}$ (see Eq. (2)). However, we can estimate them by drawing another sample $S_3$ of $m$ instance pairs, and computing a sample mean to estimate Eq. (2). According to Thm. 2, since $d_{\mathcal{D}}(A_k(S_1), A_k(S_2))$ and $d_{\mathcal{D}}(A_{k'}(S_1), A_{k'}(S_2))$ have slightly different convergence rates ($c \geq 0.01$), which are slower than $\Theta(1/m)$, then we can discern which number of clusters is more stable, with a high probability which actually *improves* as $m$ increases.

Therefore, we can use Thm. 2 as a guideline for when a stability estimator might be useful for arbitrarily large sample sizes. Namely, we need to show it is an expected value of some random variable, with at least slightly different convergence rates for different model selections, and with at least some of them dominating $\Theta(1/m)$. We would expect these conditions to hold under quite general settings, since most stability measures are based on empirically estimating the mean of some random variable. Moreover, a central-limit theorem argument leads us to expect an asymptotic form of $\Omega(1/\sqrt{m})$, with the exact constants dependent on the model. This convergence rate is slow enough for the theorem to apply. The difficult step, however, is showing that the differing convergence rates can be detected empirically, without knowledge of $\mathcal{D}$. In the example above, this reduces to showing that with high probability over $S_1$ and $S_2$, $d_{\mathcal{D}}(A_k(S_1), A_k(S_2))$ and $d_{\mathcal{D}}(A_{k'}(S_1), A_{k'}(S_2))$ will indeed differ by some constant ratio independent of $m$.

*Proof of Thm. 2.* Using a relative entropy variant of Hoeffding's bound [7], we have that for any $1 > b > 0$ and $1/\mathbb{E}[Y] > a > 1$, it holds that:

$$\Pr\left(\hat{X} \leq b\mathbb{E}[X]\right) \leq \exp\left(-m\, D_{KL}\left[b\mathbb{E}[X] \,\|\, \mathbb{E}[X]\right]\right),$$

$$\Pr\left(\hat{Y} \geq a\mathbb{E}[Y]\right) \leq \exp\left(-m\, D_{KL}\left[a\mathbb{E}[Y] \,\|\, \mathbb{E}[Y]\right]\right).$$

By substituting the bound $D_{KL}[p||q] \geq (p-q)^2/2\max\{p,q\}$ in the two inequalities, we get:

$$\Pr\left(\hat{X} \leq b\mathbb{E}[X]\right) \leq \exp\left(-\frac{1}{2}m\mathbb{E}\left[X\right](1-b)^2\right) \qquad (3)$$

$$\Pr\left(\hat{Y} \geq a\mathbb{E}[Y]\right) \leq \exp\left(-\frac{1}{2}m\mathbb{E}\left[Y\right]\left(a+\frac{1}{a}-2\right)\right), \qquad (4)$$

which hold whenever $1 > b > 0$ and $a > 1$. Let $b = 1 - (1 - \mathbb{E}[Y]/\mathbb{E}[X])^2/2$, and $a = b\mathbb{E}[X]/\mathbb{E}[Y]$. It is easily verified that $b < 1$ and $a > 1$. Substituting these values into the r.h.s of Eq. (3), and to both sides of Eq. (4), and after some algebra, we get:

$$\Pr(\hat{X} \leq b\mathbb{E}[X]) \leq \exp\left(-\frac{1}{8}m\mathbb{E}[X]\left(\frac{c}{1+c}\right)^4\right),$$

$$\Pr(\hat{Y} \geq b\mathbb{E}[X]) \leq \exp\left(-\frac{1}{4}m\mathbb{E}[X]\left(\frac{c}{1+c}\right)^2\right).$$

As a result, by the union bound, we have that $\Pr(\hat{X} \leq \hat{Y})$ is at most the sum of the r.h.s of the last two inequalities, hence proving the theorem. $\qquad\square$

As a proof of concept, we show that for a certain setting, the stability measure used by [2], as defined above, is meaningful for arbitrarily large sample sizes, even when this measure converges to zero for any choice of the required number of clusters. The result is a simple counter-example to the claim that this phenomenon makes cluster stability a problematic tool.

The setting we analyze is a mixture distribution of three well-separated unequal Gaussians in $\mathbb{R}$, where an empirical estimate of stability, using a centroid-based clustering algorithm, is utilized to discern whether the data contain $2, 3$ or $4$ clusters. We prove that with high probability, this empirical estimation process will discern $k = 3$ as much more stable than both $k = 2$ and $k = 4$ (by an amount depending on the separation between the Gaussians). The result is robust enough to hold even if in addition one performs normalization procedures to account for the fact that higher number of clusters entail more degrees of freedom for the clustering algorithm (see [9]).

We emphasize that the simplicity of this setting is merely for the sake of analytical convenience. The proof itself relies on a general and intuitive characteristic of what constitutes a 'wrong' model (namely, having cluster

boundaries in areas of high density), rather than any specific feature of this setting. We are currently working on generalizing this result, using a more involved analysis.

In this setting, by the results of [2], $stab(A_k, \mathcal{D}, m)$ will converge to 0 as $m \to \infty$ for $k = 2, 3, 4$. The next two lemmas, however, show that the stability measure for $k = 3$ (the 'correct' model order) is smaller than the other two, by a substantial ratio independent of $m$, and that this will be discerned, with high probability, based on the empirical estimates of $d_{\mathcal{D}}(A_k(S_1), A_k(S_2))$. The proofs are technical, and appear in the supplementary material to this paper.

**Lemma 1.** *For some $\mu > 0$, let $\mathcal{D}$ be a Gaussian mixture distribution on $\mathbb{R}$, with density function*

$$p(x) = \frac{2}{3\sqrt{2\pi}}\exp\left(-\frac{(x+\mu)^2}{2}\right) + \frac{1}{6\sqrt{2\pi}}\exp\left(-\frac{x^2}{2}\right) + \frac{1}{6\sqrt{2\pi}}\exp\left(-\frac{(x-\mu)^2}{2}\right).$$

*Assume $\mu \gg 1$, so that the Gaussians are well separated. Let $A_k$ be a centroid-based clustering algorithm, which is given a sample and required number of clusters $k$, and returns a set of $k$ centroids, minimizing the k-means objective function (sum of squared Euclidean distances between each instance and its nearest centroid). Then the following holds, with $o(1)$ signifying factors which converge to $0$ as $m \to \infty$:*

$$stab(A_2, \mathcal{D}, m) \geq \frac{1-o(1)}{7\sqrt{m}}\exp\left(-\frac{\mu^2}{32}\right) \quad, \quad stab(A_4, \mathcal{D}, m) \geq \frac{0.4-o(1)}{\sqrt{m}}$$

$$stab(A_3, \mathcal{D}, m) \leq \frac{1.1+o(1)}{\sqrt{m}}\exp\left(-\frac{\mu^2}{8}\right).$$

**Lemma 2.** *For the setting described in Lemma 1, it holds that over the draw of independent sample pairs $(S_1, S_2), (S_1', S_2'), (S_1'', S_2'')$ (each of size $m$ from $\mathcal{D}$), the ratio between $d_{\mathcal{D}}(A_2(S_1'), A_2(S_2'))$ and $d_{\mathcal{D}}(A_3(S_1), A_3(S_2))$, as well as the ratio between $d_{\mathcal{D}}(A_4(S_1''), A_4(S_2''))$ and $d_{\mathcal{D}}(A_3(S_1), A_3(S_2))$, is larger than 2 with probability of at least:*

$$1 - (4 + o(1)) \left( \exp\left( -\frac{\mu^2}{16} \right) + \exp\left( -\frac{\mu^2}{32} \right) \right).$$

It should be noted that the asymptotic notation is merely to get rid of second-order terms, and is not an essential feature. Also, the constants are by no means the tightest possible. With these lemmas, we can prove that a direct estimation of $stab(A, \mathcal{D}, m)$, based on a random sample, allows us to discern the more stable model with high probability, for arbitrarily large sample sizes.

**Theorem 3.** *For the setting described in Lemma 1, define the following unbiased estimator $\hat{\theta}_{k,4m}$ of $stab(A_k, \mathcal{D}, m)$: Given a sample of size $4m$, split it randomly into 3 disjoint subsets $S_1, S_2, S_3$ of size $m, m$ and $2m$ respectively. Estimate $d_{\mathcal{D}}(A_k(S_1), A_k(S_2))$ by computing*

$$\frac{1}{m} \sum_{x_i, x_{m+i} \in S_3} \mathbf{1}\Big( A_k(S_1)(x_i, x_{m+i}) \neq A_k(S_2)(x_i, x_{m+i}) \Big),$$

*where $(x_1, .., x_m)$ is a random permutation of $S_3$, and return this value as an estimate of $stab(A_k, \mathcal{D}, m)$. If three samples of size $4m$ each are drawn i.i.d from $\mathcal{D}$, and are used to calculate $\hat{\theta}_{2,4m}, \hat{\theta}_{3,4m}, \hat{\theta}_{4,4m}$, then*

$$\Pr\left( \hat{\theta}_{3,4m} \geq \min\left\{ \hat{\theta}_{2,4m}, \hat{\theta}_{4,4m} \right\} \right) \leq \exp\left( -\Omega(\mu^2) \right) + \exp\left( -\Omega\left( \sqrt{m} \right) \right).$$

*Proof.* Using Lemma 2, we have that:

$$\Pr\left( \frac{\min\left\{ d_{\mathcal{D}}(A_2(S_1'), A_2(S_2')), d_{\mathcal{D}}(A_4(S_1''), A_4(S_2'')) \right\}}{d_{\mathcal{D}}(A_3(S_1), A_3(S_2))} \leq 2 \right) < \exp\left( -\Omega(\mu^2) \right). \qquad (5)$$

Denoting the event above as $B$, and assuming it does not occur, we have that the estimators $\hat{\theta}_{2,4m}, \hat{\theta}_{3,4m}, \hat{\theta}_{4,4m}$ are each an empirical average over an additional sample of size $m$, and the expected value of $\hat{\theta}_{3,4m}$ is at least twice smaller than the expected values of the other two. Moreover, by Lemma 1, the expected value of $d_{\mathcal{D}}(A_3(S_1), A_3(S_2))$ is $\Omega(1/\sqrt{m})$. Invoking Thm. 2, we have that:

$$\Pr\left( \hat{\theta}_{3,4m} \geq \min\left\{ \hat{\theta}_{2,4m}, \hat{\theta}_{4,4m} \right\} \mid B^{\complement} \right) \leq \exp\left( -\Omega\left( \sqrt{m} \right) \right) \qquad (6)$$

Combining Eq. (5) and Eq. (6) yield the required result. $\qquad \square$

## 5   Experiments

In order to further substantiate our analysis above, some experiments were run on synthetic and real world data, with the goal of performing model selection over the number of clusters $k$. Our first experiment simulated the setting discussed in section 4 (see figure 1). We tested 3 different Gaussian mixture distributions (with $\mu = 5, 7, 8$), and sample sizes $m$ ranging from $2^5$ to $2^{22}$. For each distribution and sample size, we empirically estimated $\hat{\theta}_2$, $\hat{\theta}_3$ and $\hat{\theta}_4$ as described in section 4, using the $k$-means algorithm, and repeated this procedure over 1000 trials. Our results show that although these empirical estimators converge towards zero, their convergence rates differ, with approximately constant ratios between them. Scaling the graphs by $\sqrt{m}$ results in approximately constant and differing stability measures for each $\mu$. Moreover, the failure rate does not increase with sample size, and decreases rapidly to negligible size as the Gaussians become more well separated - exactly in line with Thm. 3. Notice that although in the previous section we assumed a large separation between the Gaussians for analytical convenience, good results are obtained even when this separation is quite small.

For the other experiments, we used the stability-based cluster validation algorithm proposed in [9], which was found to compare favorably with similar algorithms, and has the desirable property of

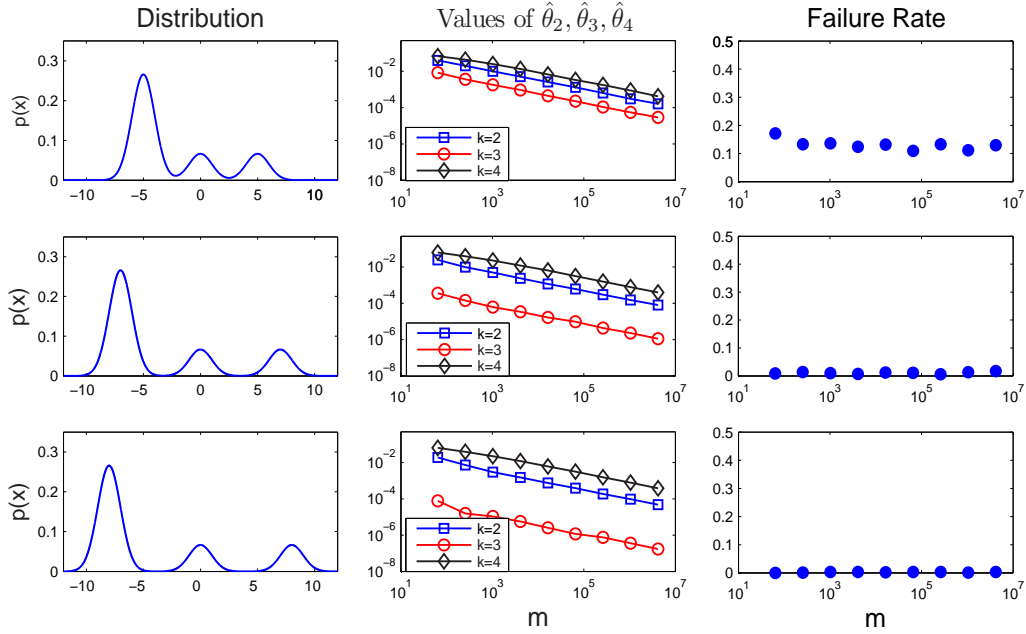

Figure 1: Empirical validation of results in section 4. In each row, the leftmost sub-figure is the actual distribution, the middle sub-figure is a log-log plot of the estimators $\hat{\theta}_2, \hat{\theta}_3, \hat{\theta}_4$ (averaged over 1000 trials), as a function of the sample size, and on the right is the failure rate as a function of the sample size (percentage of trials where $\hat{\theta}_3$ was not the smallest of the three).

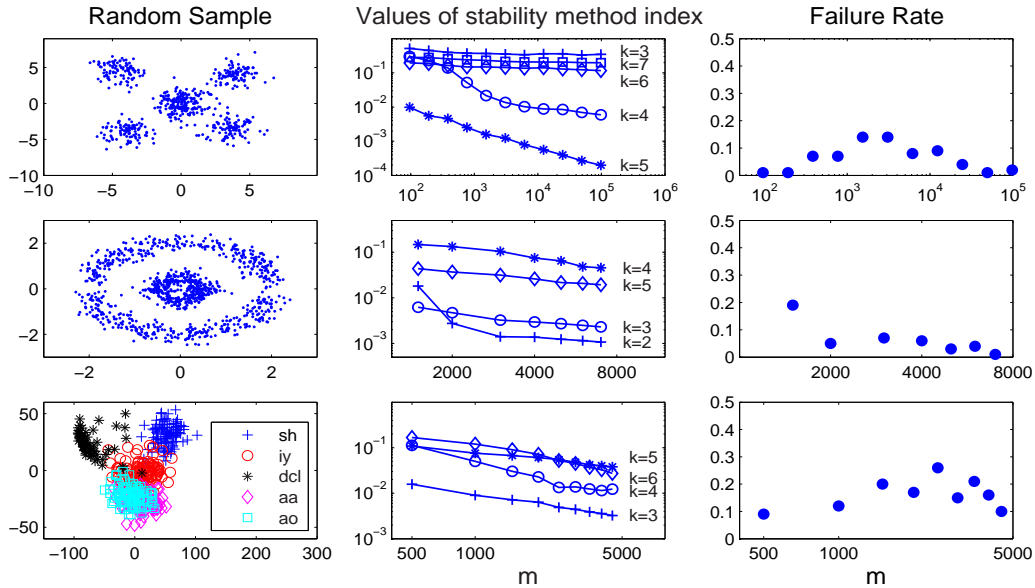

Figure 2: Performance of stability based algorithm in [9] on 3 data sets. In each row, the leftmost sub-figure is a sample representing the distribution, the middle sub-figure is a log-log plot of the computed stability indices (averaged over 100 trials), and on the right is the failure rate (in detecting the most stable model over repeated trials). In the phoneme data set, the algorithm selects 3 clusters as the most stable models, since the vowels tend to group into a single cluster. The 'failures' are all due to trials when $k = 4$ was deemed more stable.

producing a clear quantitative stability measure, bounded in $[0, 1]$. Lower values match models with higher stability. The synthetic data sets selected (see figure 2) were a mixture of 5 Gaussians, and segmented 2 rings. We also experimented on the Phoneme data set [6], which consists of $4,500$ log-periodograms of 5 phonemes uttered by English speakers, to which we applied PCA projection on 3 principal components as a pre-processing step. The advantage of this data set is its clear low-dimensional representation relative to its size, allowing us to get nearer to the asymptotic convergence rates of the stability measures. All experiments used the $k$-means algorithm, except for the ring data set which used the spectral clustering algorithm proposed in [13].

Complementing our theoretical analysis, the experiments clearly demonstrate that regardless of the actual stability measures per fixed sample size, they seem to eventually follow roughly constant and differing convergence rates, with no substantial degradation in performance. In other words, when stability works well for small sample sizes, it should also work at least as well for larger sample sizes. The universal asymptotic convergence to zero does not seem to be a problem in that regard.

## 6 Conclusions

In this paper, we propose a principled approach for analyzing the utility of stability for cluster validation in large finite samples. This approach stems from viewing stability as a measure of generalization in a statistical setting. It leads us to predict that in contrast to what might be concluded from previous work, cluster stability does not necessarily degrade with increasing sample size. This prediction is substantiated both theoretically and empirically.

The results also provide some guidelines (via Thm. 2) for when a stability measure might be relevant for arbitrarily large sample size, despite asymptotic universal stability. They also suggest that by appropriate scaling, stability measures would become insensitive to the actual sample size used. These guidelines do not presume a specific clustering framework. However, we have proven their fulfillment rigorously only for a certain stability measure and clustering setting. The proof can be generalized in principle, but only at the cost of a more involved analysis. We are currently working on deriving more general theorems on when these guidelines apply.

**Acknowledgements**: This work has been partially supported by the NATO SfP Programme and the PASCAL Network of excellence.

## Footnotes

[1]Many clustering algorithms, such as spectral clustering, do not induce a natural clustering on $\mathcal{X}$ based on a clustering of a sample. In that case, we view the algorithm as a two-stage process, in which the clustering of the sample is extended to $\mathcal{X}$ through some uniform extension operator (such as assigning instances to the 'nearest' cluster in some appropriate sense).

[2]Where we define $D_{KL}[\mathcal{Q}||\mathcal{P}] = \int_X \mathcal{Q}(X) \, ln(\mathcal{Q}(X)/\mathcal{P}(X))$, and $D_{KL}[q||p]$ for $q, p \in [0, 1]$ is defined as the divergence of Bernoulli distributions with parameters q and p.

## References

[1] Shai Ben-David. A framework for statistical clustering with a constant time approximation algorithms for k-median clustering. In *Proceedings of COLT 2004, pages 415–426.*

[2] Shai Ben-David, Ulrike von Luxburg, and Dávid Pál. A sober look at clustering stability. In *Proceedings of COLT 2006, pages 5–19.*

[3] Olivier Bousquet and André Elisseeff. Stability and generalization. *Journal of Machine Learning Research*, 2:499–526, 2002.

[4] Joachim M. Buhmann and Marcus Held. Model selection in clustering by uniform convergence bounds. In *Advances in Neural Information Processing Systems 12*, pages 216–222, 1999.

[5] Andrea Caponnetto and Alexander Rakhlin. Stability properties of empirical risk minimization over donsker classes. *Journal of Machine Learning Research*, 6:2565–2583, 2006.

[6] Trevor Hastie, Robert Tibshirani, Jerome Friedman. *The Elements of Statistical Learning*. Springer, 2001.

[7] Wassily Hoeffding. Probability inequalities for sums of bounded random variables. *Journal of the American Statistical Association*, 58(301):13–30, March 1963.

[8] Samuel Kutin and Partha Niyogi. Almost-everywhere algorithmic stability and generalization error. In *Proceeding of the 18th confrence on Uncertainty in Artificial Intelligence (UAI)*, pages 275–282, 2002.

[9] Tilman Lange, Volker Roth, Mikio L. Braun, and Joachim M. Buhmann. Stability-based validation of clustering solutions. *Neural Computation*, 16(6):1299–1323, June 2004.

[10] D.A. McAllester. Pac-bayesian stochastic model selection. *Machine Learning Journal*, 51(1):5–21, 2003.

[11] C. McDiarmid. On the method of bounded differences. In *Surveys in Combinatorics*, volume 141 of *London Mathematical Society Lecture Note Series*, pages 148–188. Cambridge University Press, 1989.

[12] Alexander Rakhlin and Andrea Caponnetto. Stability of $k$-means clustering. In *Advances in Neural Information Processing Systems 19*. MIT Press, Cambridge, MA, 2007.

[13] Jianbo Shi and Jitendra Malik. Normalized cuts and image segmentation. *IEEE Transactions on Pattern Analysis and Machine Intelligence*, 22(8):888–905, 2000.

[14] Ulrike von Luxburg and Shai Ben-David. Towards a statistical theory of clustering. Technical report, PASCAL workshop on clustering, London, 2005.

